# Probabilistic Inference of Speech Signals from Phaseless Spectrograms

**Kannan Achan,  Sam T. Roweis,  Brendan J. Frey**
Machine Learning Group
University of Toronto

## Abstract

Many techniques for complex speech processing such as denoising and deconvolution, time/frequency warping, multiple speaker separation, and multiple microphone analysis operate on sequences of short-time power spectra (spectrograms), a representation which is often well-suited to these tasks. However, a significant problem with algorithms that manipulate spectrograms is that the output spectrogram does not include a phase component, which is needed to create a time-domain signal that has good perceptual quality. Here we describe a generative model of time-domain speech signals and their spectrograms, and show how an efficient optimizer can be used to find the maximum a posteriori speech signal, given the spectrogram. In contrast to techniques that alternate between estimating the phase and a spectrally-consistent signal, our technique directly infers the speech signal, thus jointly optimizing the phase and a spectrally-consistent signal. We compare our technique with a standard method using signal-to-noise ratios, but we also provide audio files on the web for the purpose of demonstrating the improvement in perceptual quality that our technique offers.

## 1   Introduction

Working with a time-frequency representation of speech can have many advantages over processing the raw amplitude samples of the signal directly. Much of the structure in speech and other audio signals manifests itself through simultaneous common onset, offset or co-modulation of energy in multiple frequency bands, as harmonics or as coloured noise bursts. Furthermore, there are many important high-level operations which are much easier to perform in a short-time multiband spectral representation than on the time domain signal. For example, time-scale modification algorithms attempt to lengthen or shorten a signal without affecting its frequency content. The main idea is to upsample or downsample the spectrogram of the signal along the time axis while leaving the frequency axis unwarped. Source separation or denoising algorithms often work by identifying certain time-frequency regions as having high signal-to-noise or as belonging to the source of interest and "masking-out" others. This masking operation is very natural in the time-frequency domain. Of course, there are many clever and efficient speech processing algorithms for pitch tracking[6], denoising[7], and even timescale modification[4] that do operate directly on the signal samples, but the spectral domain certainly has its advantages.

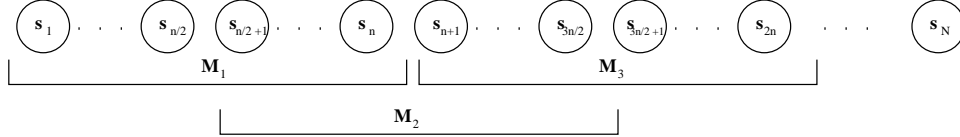

Figure 1: In the generative model, the spectrogram is obtained by taking overlapping windows of length $n$ from the time-domain speech signal, and computing the energy spectrum.

In order to reap the benefits of working with a spectrogram of the audio, it is often important to "invert" the spectral representation back into a time domain signal which is consistent with a new time-frequency representation we obtain after processing. For example, we may mask out certain cells in the spectrogram after determining that they represent energy from noise signals, or we may drop columns of the spectrogram to modify the timescale. How do we recover the denoised or sped up speech signal? In this paper we study this inversion and present an efficient algorithm for recovering signals from their overlapping short-time spectral magnitudes using maximum *a posteriori* inference in a simple probability model. This is essentially a problem of phase recovery, although with the important constraint that overlapping analysis windows must agree with each other about their estimates of the underlying waveform. The standard approach, exemplified by the classic paper of Griffin and Lim [1], is to alternate between estimating the time domain signal given a current estimate of the phase and the observed spectrogram, and estimating the phase given the hypothesized signal and the observed spectrogram. Unfortunately, at any iteration, this technique maintains inconsistent estimates of the signal and the phase.

Our algorithm maximizes the *a posteriori* probability of the estimated speech signal by adjusting the estimated signal samples directly, thus avoiding inconsistent phase estimates. At each step of iterative optimization, the method is guaranteed to reduce the discrepancy between the observed spectrogram and the spectrogram of the estimated waveform. Further, by jointly optimizing all samples simultaneously, the method can make global changes in the waveform, so as to better match all short-time spectral magnitudes.

## 2    A Generative Model of Speech Signals and Spectrograms

An advantage of viewing phase recovery as a problem of probabilistic inference of the speech signal is that a prior distribution over time-domain speech signals can be used to improve performance. For example, if the identity of the speaker that produced the spectrogram is known, a speaker-specific speech model can be used to obtain a higher-quality reconstruction of the time-domain signal. However, it is important to point out that when prior knowledge of the speaker is not available, our technique works well using a uniform prior.

For a time-domain signal with $N$ samples, let $\mathbf{s}$ be a column vector containing samples $s_1, \ldots, s_N$. We define the spectrogram of a signal as the magnitude of its windowed short-time Fourier transform. Let $\mathcal{M} = \{\mathbf{m}_1, \mathbf{m}_2, \mathbf{m}_3 \ldots\}$ denote the spectrogram of $\mathbf{s}$; $\mathbf{m}_k$ is the magnitude spectrum of the $k$th window and $m_k^f$ is the magnitude of the $f^{th}$ frequency component. Further, let $n$ be the width of the window used to obtain the short-time transform. We assume the windows are spaced at intervals of $n/2$, although this assumption is easy to relax. In this setup, shown in Fig. 1, a particular time-domain sample $s_t$ contributes to exactly two windows in the spectrogram.

The joint distribution over the speech signal $\mathbf{s}$ and the spectrogram $\mathcal{M}$ is

$$P(\mathbf{s}, \mathcal{M}) = P(\mathbf{s})P(\mathcal{M}|\mathbf{s}). \tag{1}$$

We use an $R$th-order autoregressive model for the prior distribution over time-domain speech signals:

$$P(\mathbf{s}) \propto \prod_{t=1}^{N} \exp\{-\frac{1}{2\rho^2}(\sum_{r=1}^{R} a_r s_{t-r} - s_t)^2\}. \tag{2}$$

In this model, each sample is predicted to be a linear combination of the $r$ previous samples. The autoregressive model can be estimated beforehand, using training data for a specific speaker or a general class of speakers. Although this model is overly simple for general speech signals, it is useful for avoiding discontinuities introduced at window boundaries by mis-matched phase components in neighboring frames. To avoid artifacts at frame boundaries, the variance of the prior can be set to low values at frame boundaries, enabling the prior to "pave over" the artifacts.

Assuming that the observed spectrogram is equal to the spectrogram of the hidden speech signal, plus independent Gaussian noise, the likelihood can be written

$$P(\mathcal{M}|\mathbf{s}) \propto \prod_{k} \exp\{-\frac{1}{2\sigma^2}||\hat{\mathbf{m}}_k(s) - \mathbf{m}_k||^2\} \tag{3}$$

where $\sigma^2$ is the noise in the observed spectra, and $\hat{\mathbf{m}}_k(s)$ is the magnitude spectrum given by the appropriate window of the estimated speech signal, $\mathbf{s}$. Note that the magnitude spectra are independent given the time domain signal.

The likelihood in (3) favors configurations of $\mathbf{s}$ that match the observed spectrogram, while the prior in (2) places more weight on configurations that match the autoregressive model.

## 2.1 Making the speech signal explicit in the model

We can simplify the functional form $\hat{\mathbf{m}}_k(s)$, by introducing the $n \times n$ Fourier transform matrix, $\mathbf{F}$. Let $\mathbf{s}_k$ be an $n$-vector containing the samples from the $k$th window. Using the fact that the magnitude of a complex number $c$ is $cc^*$, where $^*$ denotes complex conjugation, we have

$$\hat{\mathbf{m}}_k(\mathbf{s}) = (\mathbf{F}\mathbf{s}_k) \circ (\mathbf{F}\mathbf{s}_k)^* = (\mathbf{F}\mathbf{s}_k) \circ (\mathbf{F}^*\mathbf{s}_k),$$

where $\circ$ indicates element-wise product.

The joint distribution in (1) can now be written

$$P(\mathbf{s}, \mathcal{M}) \propto \prod_{k} \exp\{-\frac{1}{2\sigma^2}||(\mathbf{F}\mathbf{s}_k) \circ (\mathbf{F}^*\mathbf{s}_k) - \mathbf{m}_k||^2\} \prod_{t} \exp\{-\frac{1}{2\rho^2}(\sum_{r=1}^{R} a_r s_{t-r} - s_t)^2\}. \tag{4}$$

The factorization of the distribution in (4) can be used to construct the factor graph shown in Fig. 2. For clarity, we have used a 3rd order autoregressive model and a window length of 4. In this graphical model, function nodes are represented by black disks and each function node corresponds to a term in the joint distribution. There is one function node connecting each observed short-time energy spectrum to the set of $n$ time-domain samples from which it was possibly derived, and one function node connecting each time-domain sample to its $R$ predecessors in the autoregressive model.

Taking the logarithm of the joint distribution in (4) and expanding the norm, we obtain

$$\log P(\mathbf{s}, \mathcal{M}) \propto -\frac{1}{2\sigma^2} \sum_{k} \sum_{i} \Big( \sum_{j=1}^{n} \sum_{l=1}^{n} F_{ij} F_{il}^* s_{nk-n/2+j} s_{nk-n/2+l} - m_{ki} \Big)^2$$

$$-\frac{1}{2\rho^2} \sum_{t} \Big( \sum_{r=1}^{R} a_r s_{t-r} - s_t \Big)^2. \tag{5}$$

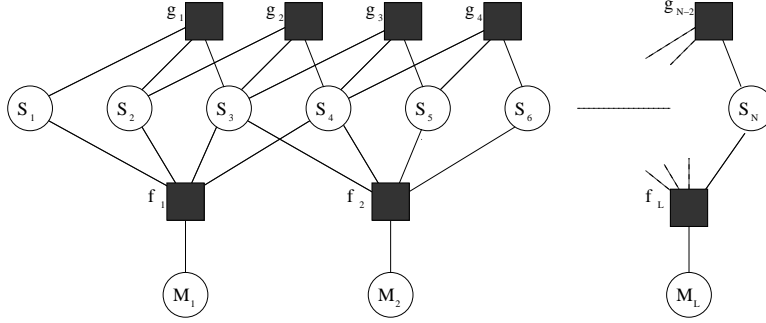

Figure 2: Factor graph for the model in (4) using a $3^{rd}$ order autoregressive model, window length of 4 and an overlap of 2 samples. Function nodes $f_i$ enforce the constraint that the spectrogram of **s** match the observed spectrogram and function nodes $g_i$ enforce the constraint due to the AR model

In this expression, $k$ indexes frames, $i$ indexes frequency, $s_{k-n/2+j}$ is the $j$th sample in the $k$th frame, $m_{ki}$ is the observed spectral energy at frequency $i$ in frame $k$, and $a_r$ is the $r$th autoregressive coefficient. The log-probability is quartic in the unknown speech samples, $s_1, \ldots, s_N$.

For simplicity of presentation above, we implicitly assumed a rectangular window for computing the spectrogram. The extension to other types of windowing functions is straightforward. In the experiments described below, we have used a Hamming window, and adjusted the equations appropriately.

## 3   Inference Algorithms

The goal of probabilistic inference is to compute the posterior distribution over speech waveforms and output a typical sample or a mode of the posterior as an estimate of the reconstructed speech signal. To find a mode of the posterior, we have explored the use of iterative conditional modes (ICM) [8], Markov chain Monte Carlo methods [9], variational techniques [10], and direct application of numerical optimization methods for finding the maximum *a posteriori* speech signal. In this paper, we report results on two of the faster techniques, ICM and direct optimization.

ICM operates by iteratively selecting a variable and assigning the MAP estimate to the variable while keeping all other variables fixed. This technique is guaranteed to increase the joint probability of the speech waveform and the observed spectrum, at each step. At every stage we set $s_t$ to its most probable value, given the other speech samples and the observed spectrogram:

$$s_t^* = \mathrm{argmax}_{s_t} P(s_t | \mathcal{M}, \mathbf{s} \setminus s_t) = \mathrm{argmax}_{s_t} P(\mathbf{s}, \mathcal{M}).$$

This value can be found by extracting the terms in (5) that depend on $s_t$ and optimizing the resulting quartic equation with complex coefficients. To select an initial configuration of **s**, we applied an inverse Fourier transform to the observed magnitude spectra $\mathcal{M}$, assuming a random phase. As will become evident in the experimental section of this paper, by updating only a single sample at a time, ICM is prone to finding poor local minima.

We also implemented an inference algorithm that directly searches for a maximum of $\log P(\mathbf{s}, \mathcal{M})$ w.r.t. **s**, using conjugate gradients. The same derivatives used to find the ICM updates were used in a conjugate gradient optimizer, which is capable of finding search directions in the vector space **s**, and *jointly* adjusting all speech samples simultaneously. We

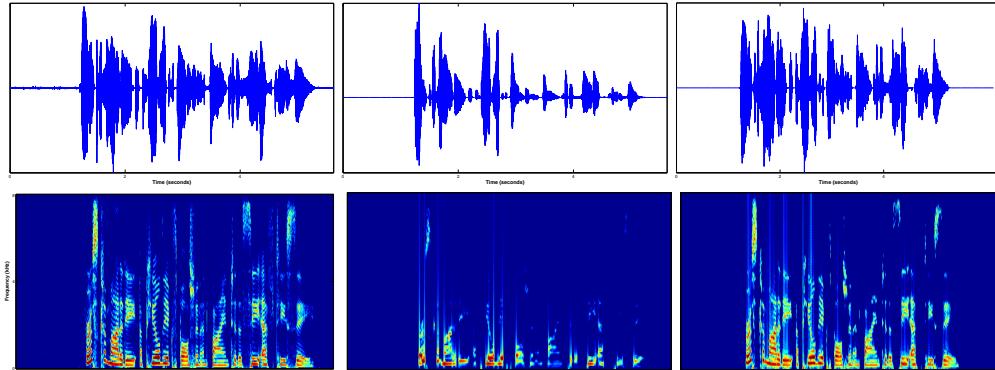

Figure 3: Reconstruction results for an utterance from the WSJ database. (*left*) Original signal and the corresponding spectrogram. (*middle*) Reconstruction using algorithm in [1]. The spectrogram of the reconstruction fails to capture the finer details in the original signal. (*right*) Reconstruction using our algorithm. The spectrogram captures most of the fine details in the original signal.

initialized the conjugate gradient optimizer using the same procedure as described above for ICM.

## 4   Experiments

We tested our algorithm using several randomly chosen utterances from the Wall street journal corpus and the NIST TIMIT corpus. For all experiments we used a (Hamming) window of length 256 and with an overlap of 128 samples. Where possible, we trained a $12^{th}$ order AR model of the speaker using an utterance different from the one used to create the spectrogram. For convergence to a good local minima, it is important to down weight the contribution of the AR-model for the first several iterations of conjugate gradient optimization. In fact we ran the algorithm without the AR model until convergence and then started the AR model with a weighting factor of 10. This way, the AR model operates on the signal with very little error in the estimated spectrogram.

Along the frame boundaries, the variance of the prior (AR model) was set to a small value to smooth down spikes that are not very probable *apriori*. Further, we also tried using a cubic spline smoother along the boundaries as a post processing step for better sound quality.

### 4.1   Evaluation

The quality of sound in the estimated signal is an important factor in determining the effectiveness of the algorithm. To demonstrate improvement in the perceptual quality of sound we have placed audio files on the web; for demonstrations please check, http://www.psi.toronto.edu/~kannan/spectrogram. Our algorithm consistently outperformed the algorithm proposed in [1] both in terms of sound quality and in matching the observed spectrogram . Fig. 3 shows reconstruction result for an utterance from WSJ data.

As expected, ICM typically converged to a poor local minima in a few iterations. In Fig. 4, a plot of the log probability as a function of number of iterations is shown for ICM and our approach.

| Algorithm | dB gain (dB) |
|---|---|
| Griffin and Lim [1] | 4.508 |
| Our approach (without AR model) | 7.900 |
| Our approach ($12^{th}$ order AR model) | 8.172 |

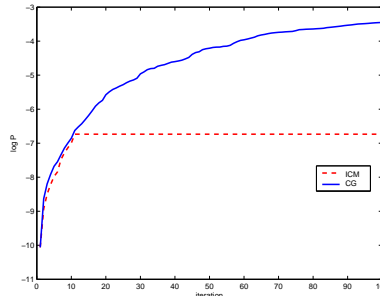

Figure 4: SNR for different algorithms. Values reported are averages over 12 different utterances. The graph on the right compares the log probability under ICM to our algorithm

Analysis of signal to noise ratio of the true and estimated signal can be used to measure the quality of the estimated signal, with high dB gain indicating good reconstruction.

As the input to our model does not include a phase component, we cannot measure SNR by comparing the recovered signal to any true time domain signal. Instead, we define the following approximation

$$\mathcal{SNR}^* = \sum_u 10 \log \frac{\frac{1}{E_u} \sum_w \sum_f |s_{u,w}(f)|^2}{\sum_w \sum_f (\frac{1}{E_u}|\hat{s}_{u,w}(f)| - \frac{1}{E_u}|s_{u,w}(f)|)^2} \tag{6}$$

where $E_u = \sum_t s_t^2$ is the total energy in utterance $u$. Summations over $u$, $w$ and $f$ are over all utterances, windows and frequencies respectively.

The table in Fig. 4 reports dB gain averaged over several utterances for [1] and our algorithm with and without an AR model.The gains for our algorithm are significantly better than for the algorithm of Griffin and Lim. Moving the summation over $w$ in (6) outside the log produces similar quality estimates.

### 4.2 Time Scale Modification

As an example to show the potential utility of spectrogram inversion, we investigated an extremely simple approach to time scale modification of speech signals. Starting from the original signal we form the spectrogram (or else we may start with the spectrogram directly), and upsample or downsample it along the time axis. (For example, to speed up the speech by a factor of two we can discard every second column of the spectrogram.) In spite of the fact that this approach does not use any phase information from the original signal, it produces results with good perceptual sound quality. (Audio demonstrations are available on the web site given earlier.)

## 5 Variational Inference

The framework described so far focuses on obtaining fixed point estimates for the time domain signal by maximizing the joint log probability of the model in (5). A more important and potentially useful task is to find the posterior probability distribution $P(\mathbf{s}|\mathcal{M})$. As exact inference of $P(\mathbf{s}|\mathcal{M})$ is intractable, we approximate it using a fully factored distribution $Q(\mathbf{s})$ where,

$$Q(\mathbf{s}) = \prod_i q_i(s_i) \tag{7}$$

Here we assume $q_i(s_i) \sim \mathcal{N}(\mu_i, \eta_i)$. The goal of variational approximation is to infer the parameters $\{\mu_i, \eta_i\}, \forall i$ by minimizing the KL divergence between the approximating $\mathbf{Q}$ distribution and the true posterior $P(\mathbf{s}|\mathcal{M})$. This is equivalent to minimizing,

$$
\begin{aligned}
D &= \sum_s Q(\mathbf{s}) \log \frac{Q(\mathbf{s})}{P(\mathbf{s}, \mathcal{M})} \\
&= \sum_s (\prod_i q_i(s_i)) \log \frac{(\prod_i q_i(s_i))}{P(\mathbf{s}, \mathcal{M})} \\
&= -\sum_i \mathcal{H}(q_i) - \mathcal{E}_Q(\log P(\mathbf{s}, \mathcal{M}))
\end{aligned} \tag{8}
$$

The entropy term $\mathcal{H}(q_i)$ is easy to compute; $\log P(\mathbf{s}, \mathcal{M})$ is a quartic in the random variable $s_i$ and the second term involves computing the expectation of it with respect to the $\mathbf{Q}$ distribution. Simplifying and rearranging terms we get,

$$
\begin{aligned}
D &= -\sum_i \mathcal{H}(q_i) - \Big(\sum_{j=1}^n \sum_{l=1}^n F_{ij} F_{il}^* \mu_{nk-n/2+j} \mu_{nk-n/2+l} - m_{ki}\Big)^2 \\
&\quad + \sum_i \eta_i^2 G_i(\mu, \eta)
\end{aligned} \tag{9}
$$

$G_i(\mu, \eta)$ accounts for uncertainty in $\mathbf{s}$. Estimates with high uncertainty ($\eta$) will tend to have very little influence on other estimates during the optimization. Another interesting aspect of this formulation is that by setting $\eta = 0$, the first and third terms in (9) vanish and $D$ takes a form similar to (5). In other words, in the absence of uncertainty we are in essence finding fixed point estimates for $\mathbf{s}$.

## 6  Conclusion

In this paper, we have introduced a simple probabilistic model of noisy spectrograms in which the samples of the unknown time domain signal are represented directly as hidden variables. But using a continuous gradient optimizer on these quantities, we are able to accurately estimate the full speech signal from only the short time spectral magnitudes taken in overlapping windows. Our algorithm's reconstructions are substantially better, both in terms of informal perceptual quality and measured signal to noise ratio, than the standard approach of Griffin and Lim[1]. Furthermore, in our setting, it is easy to incorporate an a-priori model of gross speech structure in the form of an AR-model, whose influence on the reconstruction is user-tunable. Spectrogram inversion has many potential applications; as an example we have demonstrated an extremely simple but nonetheless effective time scale modification algorithm which subsamples the spectrogram of the original utterance and then inverts.

In addition to improved experimental results, our approach highlights two important lessons from the point of view of statistical signal processing algorithms. The first is that directly representing quantities of interest and making inferences about them using the machinery of probabilistic inference is a powerful approach that can avoid the pitfalls of less principled

iterative algorithms that maintain inconsistent estimates of redundant quantities, such as phase and time-domain signals. The second is that coordinate descent optimization (ICM) does not always yield the best results in problems with highly dependent hidden variables. It is often tacitly assumed in the graphical models community, that the more structured an approximation one can make when updating blocks of parameters simultaneously, the better. In other words, practitioners often try to solve for as more variables as possible conditioned on quantities that have just been updated. Our experience in this model has shown that direct continuous optimization using gradient techniques allows all quantities to adjust simultaneously and ultimately finds far superior solutions.

Because of its probabilistic nature, our model can easily be extended to include other pieces of prior information, or to deal with missing or noisy spectrogram frames. This opens the door to unified phase recovery and denoising algorithms, and to the possibility of performing sophisticated speech separation or denoising inside the pipeline of a standard speech recognition system, in which typically only short time spectral magnitudes are available.

**Acknowledgments**

We thank Carl Rasmussen for his conjugate gradient optimizer. KA, STR and BJF are supported in part by the Natural Sciences and Engineering Research Council of Canada. BJF and STR are supported in part by the Ontario Premier's Research Excellence Award. STR is supported in part by the Learning Project of IRIS Canada.

# References

[1] Griffin, D. W and Lim, J. S Signal estimation from modified short time Fourier transform In *IEEE Transactions on Acoustics, Speech and Signal Processing, 1984* 32/2

[2] Kschischang, F. R., Frey, B. J. and Loeliger, H. A. Probability propagation and iterative decoding.Factor graphs and the sum-product algorithm In *IEEE Transactions on Information Theory, 2001* 47

[3] Fletcher, R *Practical methods of optimization* . John Wiley & Sons, 1987.

[4] Roucos, S. and A. M. Wilgus. High Quality Time-Scale Modification for Speech. *In Proceedings of the International Conference on Acoustics, Speech, and Signal Processing, IEEE, 1985, 493-496.*

[5] Rabiner, L. and Juang, B. Fundamentals of Speech Recognition. *Prentice Hall, 1993*

[6] L. K. Saul, D. D. Lee, C. L. Isbell, and Y. LeCun Real time voice processing with audiovisual feedback: toward autonomous agents with perfect pitch. *in S. Becker, S. Thrun, and K. Obermayer (eds.), Advances in Neural Information Processing Systems 15. MIT Press: Cambridge, MA, 2003*

[7] Eric A. Wan and Alex T. Nelson Removal of noise from speech using the dual EKF algorithm *in Proceedings of the International Conference on Acoustics, Speech, and Signal Processing (ICASSP), IEEE, May, 1998*

[8] Besag, J On the statistical analysis of dirty pictures *Journal of the Royal Statistical Society B vol.48, pg 259–302, 1986*

[9] Neal, R. M, Probabilistic inference using Markov chain Monte Carlo Methods, *University of Toronto Technical Report 1993*

[10] M. I. Jordan and Z. Ghahramani and T. S. Jaakkola and L. K. Saul An introduction to variational methods for graphical models *Learning in Graphical Models, edited by M. I. Jordan, Kluwer Academic Publishers, Norwell MA., 1998.*
